# Adding Constrained Discontinuities to Gaussian Process Models of Wind Fields

**Dan Cornford***     **Ian T. Nabney**     **Christopher K. I. Williams[†]**
Neural Computing Research Group
Aston University, BIRMINGHAM, B4 7ET, UK
d.cornford@aston.ac.uk

## Abstract

Gaussian Processes provide good prior models for spatial data, but can be too smooth. In many physical situations there are discontinuities along bounding surfaces, for example fronts in near-surface wind fields. We describe a modelling method for such a constrained discontinuity and demonstrate how to infer the model parameters in wind fields with MCMC sampling.

## 1  INTRODUCTION

We introduce a model for wind fields based on Gaussian Processes (GPs) with 'constrained discontinuities'. GPs provide a flexible framework for modelling various systems. They have been adopted in the neural network community and are interpreted as placing priors over functions.

Stationary vector-valued GP models (Daley, 1991) can produce realistic wind fields when run as a generative model; however, the resulting wind fields do not contain some features typical of the atmosphere. The most difficult features to include are surface fronts. Fronts are generated by complex atmospheric dynamics and are marked by large changes in the surface wind direction (see for example Figures 2a and 3b) and temperature. In order to account for such features, which appear discontinuous at our observation scale, we have developed a model for vector-valued GPs with constrained discontinuities which could also be applied to surface reconstruction in computer vision, and geostatistics.

In section 2 we illustrate the generative model for wind fields with fronts. Section 3 explains what we mean by GPs with constrained discontinuities and derives the likelihood of data under the model. Results of Bayesian estimation of the model parameters are given,

[†]Now at: Division of Informatics, University of Edinburgh, 5 Forrest Hill, Edinburgh EH1 2QL, Scotland, UK

using a Markov Chain Monte Carlo (MCMC) procedure. In the final section, the strengths and weaknesses of the model are discussed and improvements suggested.

## 2  A GENERATIVE WIND FIELD MODEL

We are primarily interested in retrieving wind fields from satellite scatterometer observations of the ocean surface[1]. A probabilistic prior model for wind fields will be used in a Bayesian procedure to resolve ambiguities in local predictions of wind direction. The generative model for a wind field including a front is taken to be a combination of two vector-valued GPs with a constrained discontinuity.

A common method for representing wind fields is to put GP priors over the velocity potential $\Phi$ and stream function $\Psi$, assuming the processes are uncorrelated (Daley, 1991). The horizontal wind vector $\boldsymbol{u} = (u, v)$ can then be derived from:

$$u = -\frac{\partial \Psi}{\partial y} + \frac{\partial \Phi}{\partial x}, \qquad v = \frac{\partial \Psi}{\partial x} + \frac{\partial \Phi}{\partial y}. \qquad (1)$$

This produces good prior models for wind fields when a suitable choice of covariance function for $\Phi$ and $\Psi$ is made. We have investigated using a modified Bessel function based covariance[2] (Handcock and Wallis, 1994) but found, using three years of wind data for the North Atlantic, that the maximum *a posteriori* value for the smoothness parameter[3] in this covariance function was $\sim 2.5$. Thus we used the correlation function:

$$\rho(r) = \left(1 + \frac{r}{L} + \frac{r^2}{3L^2}\right) \exp\left(-\frac{r}{L}\right) \qquad (2)$$

where $L$ is the correlation length scale, which is equivalent to the modified Bessel function and less computationally demanding (Cornford, 1998).

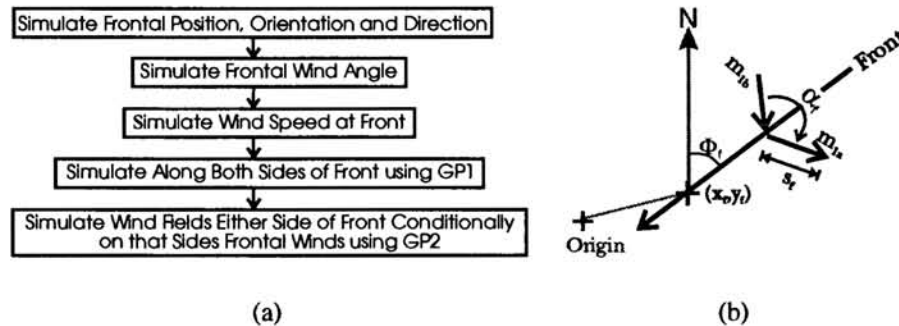

|     (a)     |     (b)     |

Figure 1: (a) Flowchart describing the generative frontal model. See text for full description. (b) A description of the frontal model.

The generative model has the form outlined in Figure 1a. Initially the frontal position and orientation are simulated. They are defined by the angle clockwise from north ($\phi_f$) that the front makes and a point on the line ($x_f, y_f$). Having defined the position of the front,

for details of the scatterometer work. Technical reports describing, in more detail, methods for generating prior wind field models can also be accessed from the same page.

[2]The modified Bessel function allows us to control the differentiability of the sample realisations through the 'smoothness parameter', as well as the length scales and variances.

[3]This varies with season, but is the most temporally stable parameter in the covariance function.

the angle of the wind across the front ($\alpha_f$) is simulated from a distribution covering the range $[0, \pi)$. This angle is related to the vertical component of vorticity ($\zeta$) across the front through $\zeta = k \cdot \nabla \times \boldsymbol{u} \propto \cos\left(\frac{\alpha_f}{2}\right)$ and the constraint $\alpha_f \in [0, \pi)$ ensures cyclonic vorticity at the front. It is assumed that the front bisects $\alpha_f$. The wind speed ($s_f$) is then simulated at the front. Since there is generally little change in wind speed across the front, one value is simulated for both sides of the front. These components $\boldsymbol{\theta_f} = (\phi_f, x_f, y_f, \alpha_f, s_f)$ define the line of the front and the mean wind vectors just ahead of and just behind the front (Figure 1b):

$$m_{1a} = (u_{1a}^m, v_{1a}^m) = \left(s_f \sin\left(\phi_f + \frac{\alpha_f}{2}\right), s_f \cos\left(\phi_f + \frac{\alpha_f}{2}\right)\right) \tag{3}$$

$$m_{1b} = (u_{1b}^m, v_{1b}^m) = \left(-s_f \sin\left(\phi_f - \frac{\alpha_f}{2}\right), -s_f \cos\left(\phi_f - \frac{\alpha_f}{2}\right)\right) \tag{4}$$

A realistic model requires some variability in wind vectors along the front. Thus we use a GP with a non-zero mean ($m_{1a}$ or $m_{1b}$) along the line of the front. In the real atmosphere we observe a smaller variability in the wind vectors along the line of the front compared with regions away from fronts. Thus we use different GP parameters along the front ($GP_1$), from those used in the wind field away from the front ($GP_2$), although the same $GP_1$ parameters are used both sides of the front, just with different means. The winds just ahead of and behind the front are assumed conditionally independent given $m_{1a}$ and $m_{1b}$, and are simulated at a regular $50~km$ spacing. The final step in the generative model is to simulate wind vectors using $GP_2$ in both regions either side of the front, conditionally on the values along that side of the front. This model is flexible enough to represent fronts, yet has the required constraints derived from meteorological principles, for example that fronts should always be associated with cyclonic vorticity and that discontinuities at the model scale should be in wind direction but not in wind speed[4]. To make this generative model useful for inference, we need to be able to compute the data likelihood, which is the subject of the next section.

## 3  GPs WITH CONSTRAINED DISCONTINUITIES

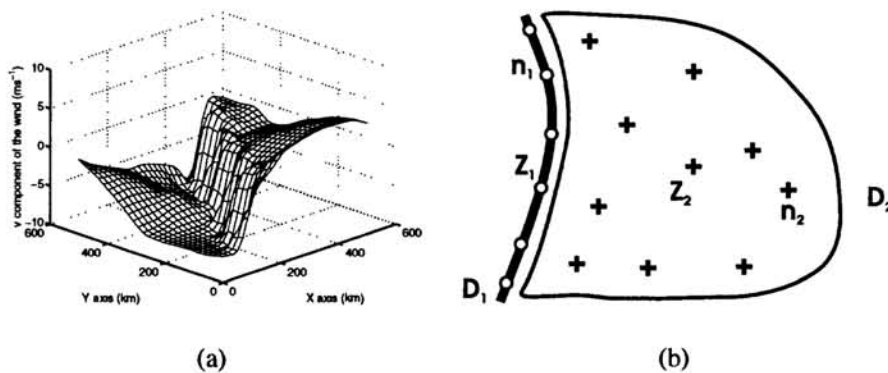

(a)                                        (b)

Figure 2: (a) The discontinuity in one of the vector components in a simulation. (b) Framework for GPs with boundary conditions. The curve $D_1$ has $n_1$ sample points with values $Z_1$. The domain $D_2$ has $n_2$ points with values $Z_2$.

We consider data from two domains $D_1$ and $D_2$ (Figure 2b), where in this case $D_1$ is a curve in the plane which is intended to be the front and $D_2$ is a region of the plane. We obtain $n_1$ variables $Z_1$ at points $x_1$ along the curve, and we assume these are generated under $GP_1$ (a GP which depends on parameters $\theta_1$ and has mean $m_1 = m_1 \mathbf{1}$ which will be determined by (3) or (4)). We are interested in determining the likelihood of the variables $Z_2$ observed at $n_2$ points $x_2$ under $GP_2$ which depends on parameters $\theta_2$, conditioned on the 'constrained discontinuities' at the front.

We evaluate this by calculating the likelihood of $Z_2$ conditioned on the $n_1$ values of $Z_1$ from $GP_1$ along the front and marginalising out $Z_1$:

$$p(Z_2|\theta_2,\theta_1) = \int_{-\infty}^{\infty} p(Z_2|Z_1,\theta_2,\theta_1,m_1)p(Z_1|\theta_1,m_1)\,dZ_1. \tag{5}$$

From the definition of the likelihood of a GP (Cressie, 1993) we find:

$$p(Z_2|Z_1,\theta_2,\theta_1,m_1) = \frac{1}{(2\pi)^{\frac{n_2}{2}}|S_{22}|^{\frac{1}{2}}} \exp\left(-\frac{1}{2}Z_2^{*\prime}S_{22}^{-1}Z_2^{*}\right) \tag{6}$$

where:

$$S_{22} = K_{22|2} - K'_{12|2}K_{11|2}^{-1}K_{12|2}, \qquad Z_2^{*} = Z_2 - K'_{12|2}K_{11|2}^{-1}Z_1.$$

To understand the notation consider the joint distribution of $Z_1, Z_2$ and in particular its covariance matrix:

$$K = \begin{bmatrix} K_{11|2} & K_{12|2} \\ K_{21|2} & K_{22|2} \end{bmatrix} \tag{7}$$

where $K_{11|2}$ is the $n_1 \times n_1$ covariance matrix between the points in $D_1$ evaluated using $\theta_2$, $K_{12|2} = K'_{21|2}$ the $n_1 \times n_2$ (cross) covariance matrix between the points in $D_1$ and $D_2$ evaluated using $\theta_2$ and $K_{22|2}$ is the usual $n_2 \times n_2$ covariance for points in $D_2$. Thus we can see that $S_{22}$ is the $n_2 \times n_2$ modified covariance for the points in $D_2$ given the points along $D_1$, while the $Z_2^{*}$ is the corrected mean that accounts for the values at the points in $D_1$, which have non-zero mean.

We remove the dependency on the values $Z_1$ by evaluating the integral in (5). $p(Z_1|\theta_1,m_1)$ is given by:

$$p(Z_1|\theta_1,m_1) = \frac{1}{(2\pi)^{\frac{n_1}{2}}|K_{11|1}|^{\frac{1}{2}}} \exp\left(-\frac{1}{2}\left(Z_1 - m_1\right)' K_{11|1}^{-1}\left(Z_1 - m_1\right)\right) \tag{8}$$

where $K_{11|1}$ is the $n_1 \times n_1$ covariance matrix between the points in $D_1$ evaluated under the covariance given by $\theta_1$. Completing the square in $Z_1$ in the exponent, the integral (5) can be evaluated to give:

$$p(Z_2|\theta_2,\theta_1,m_1) = \frac{1}{(2\pi)^{\frac{n_2}{2}}} \frac{1}{|S_{22}|^{\frac{1}{2}}} \frac{1}{|K_{11|1}|^{\frac{1}{2}}} \frac{1}{|B|^{\frac{1}{2}}} \times \tag{9}$$

$$\exp\left(\frac{1}{2}\left(C'B^{-1}C - Z_2'S_{22}^{-1}Z_2 - m_1'K_{11|1}^{-1}m_1\right)\right)$$

where:

$$B = (K'_{12|2}K_{11|2}^{-1})'S_{22}^{-1}K'_{12|2}K_{11|2}^{-1} + K_{11|1}^{-1}$$

$$C' = Z_2'S_{22}^{-1}K'_{12|2}K_{11|2}^{-1} + m_1'K_{11|1}^{-1}$$

The algorithm has been coded in MATLAB and can deal with reasonably large numbers of points quickly. For a two dimensional vector-valued GP with $n_1 = 12$ and $n_2 = 200$ [5] and

a covariance function given by (2), computation of the log likelihood takes 4.13 seconds on an SGI Indy R5000.

The mean value just ahead and behind the front define the mean values for the constrained discontinuity (i.e. $m_1$ in (9)). Conditional on the frontal parameters the wind fields either side (Figure 3a) are assumed independent:

$$p(Z_{2a}, Z_{2b}|\theta_2, \theta_1, \theta_f) = p(Z_{2a}|\theta_2, \theta_1, m_{1a})p(m_{1a}|\theta_f) \times$$
$$p(Z_{2b}|\theta_2, \theta_1, m_{1b})p(m_{1b}|\theta_f)$$

where we have performed the integration (5) to remove the dependency on $Z_{1a}$ and $Z_{1b}$. Thus the likelihood of the data $Z_2 = (Z_{2a}, Z_{2b})$ given the model parameters $\theta_2, \theta_1, \theta_f$ is simply the product of the likelihoods of two GPs with a constrained discontinuity which can be computed using (9).

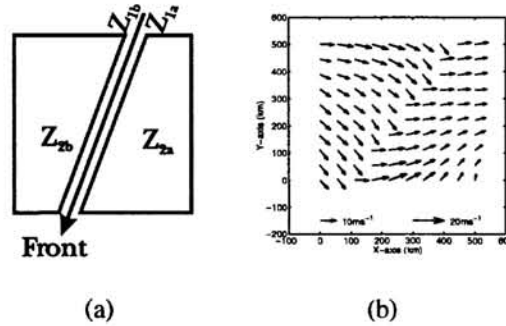

(a)                    (b)

Figure 3: (a) The division of the wind field using the generative frontal model. $Z_{1a}$, $Z_{1b}$ are the wind fields just ahead and behind the front, along its length, respectively. $Z_{2a}$, $Z_{2b}$ are the wind fields in the regions ahead of and behind the front respectively. (b) An example from the generative frontal model: the wind field looks like a typical 'cold front'.

The model outlined above was tested on simulated data generated from the model to assess parameter sensitivity. We generated a wind field $Z^o = (Z_{2a}^o, Z_{2b}^o)$ using known model parameters (e.g. Figure 3b). We then sampled the model parameters from the posterior distribution:

$$p(\theta_2, \theta_1, \theta_f|Z^o) \propto p(Z^o|\theta_2, \theta_1, \theta_f)p(\theta_2)p(\theta_1)p(\theta_f) \qquad (10)$$

where $p(\theta_2), p(\theta_1), p(\theta_f)$ are prior distributions over the parameters in the GPs and front models. This brings out one advantage of the proposed model. All the model parameters have a physical interpretation and thus expert knowledge was used to set priors which produce realistic wind fields. We will also use (10) to help set (hyper)priors using real data in $Z^o$.

MCMC using the Metropolis algorithm (Neal, 1993) is used to sample from (10) using the NETLAB[6] library. Convergence of the Markov chain is currently assessed using visual inspection of the univariate sample paths since the generating parameters are known, although other diagnostics could be used (Cowles and Carlin, 1996). We find that the procedure is insensitive to the initial value of the GP parameters, but that the parameters describing the location of the front ($\phi_f, d_f$) need to be initialised 'close' to the correct values if the chain is to converge on a reasonable time-scale. In the application some preliminary analysis of the wind field would be necessary to identify possible fronts and thus set the initial parameters to 'sensible' values. We intend to fit a vector-valued GP without any discontinuities

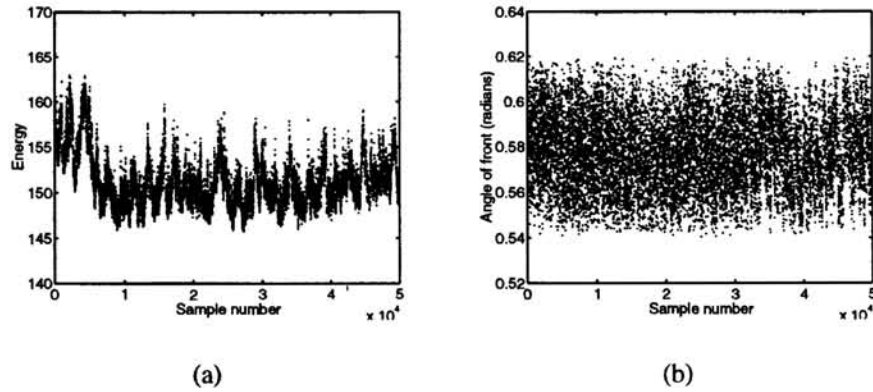

(a)                                                              (b)

Figure 4: Examples from the Markov chain of the posterior distribution (10). (a) The energy = negative log posterior probability. Note that the energy when the chain was initialised was 2789 and the first 27 values are outside the range of the y-axis. (b) The angle of the front relative to north ($\phi_f$).

and then measure the 'strain' or misfit of the locally predicted winds with the winds fitted by the GP. Lines of large 'strain' will be used to initialise the front parameters.

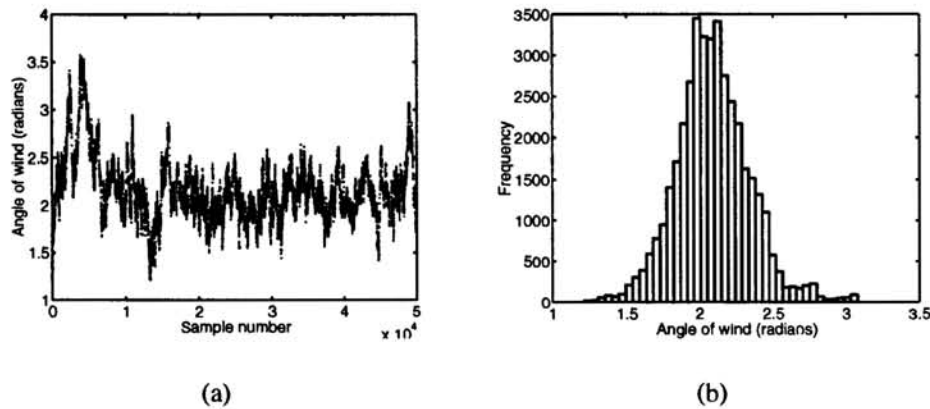

(a)                                                              (b)

Figure 5: Examples from the Markov chain of the posterior distribution (10). (a) The angle of the wind across the front ($\alpha_f$). (b) Histogram of the posterior distribution of $\alpha_f$ allowing a 10000 iteration burn-in period.

Examples of samples from the Markov chain from the simulated wind field shown in Figure 3a can be seen in Figures 4 and 5. Figure 4a shows that the energy level (= negative log posterior probability) falls very rapidly to near its minimum value from its large starting value of 2789. In these plots the true parameters for the front were $\phi_f = 0.555, \alpha_f = 2.125$ while the initial values were set at $\phi_f = 0.89, \alpha_f = 1.49$. Other parameters were also incorrectly set. The Metropolis algorithm seems to be able to find the minimum and then stays in it.

Figure 4b and 5a show the Markov chains for $\phi_f$ and $\alpha_f$. Both converge quickly to an apparently stationary distributions, which have mean values very close to the 'true' generating parameters. The histogram of the distribution of $\alpha_f$ is shown in Figure 5b.

## 4  DISCUSSION AND CONCLUSIONS

Simulations from our model are meteorologically plausible wind fields which contain fronts. It is possible similar models could usefully be applied to other modelling problems where there are discontinuities with known properties. A method for the computation of the likelihood of data given two GP models, one with non-zero mean on the boundary and another in the domain in which the data is observed, has been given. This allows us to perform inference on the parameters in the frontal model using a Bayesian approach of sampling from the posterior distribution using a MCMC algorithm.

There are several weaknesses in the model specifically for fronts, which could be improved with further work. Real atmospheric fronts are not straight, thus the model would be improved by allowing 'curved' fronts. We could represent the position of the front, oriented along the angle defined by $\phi_f$ using either another smooth GP, B-splines or possibly polynomials.

Currently the points along the line of the front are simulated at the mean observation spacing in the rest of the wind field ($\sim 50 \ km$). Interesting questions remain about the (in-fill) asymptotics (Cressie, 1993) as the distance between the points along the front tends to zero. Empirical evidence suggests that as long as the spacing along the front is 'much less' than the length scale of the GP along the front (which is typically $\sim 1000 \ km$) then the spacing does not significantly affect the results.

Although we currently use a Metropolis algorithm for sampling from the Markov chain, the derivative of (9) with respect to the GP parameters $\theta_1$ and $\theta_2$ could be computed analytically and used in a hybrid Monte Carlo procedure (Neal, 1993).

These improvements should lead to a relatively robust procedure for putting priors over wind fields which will be used with real data when retrieving wind vectors from scatterometer observations over the ocean.

### Acknowledgements

This work was partially supported by the European Union funded NEUROSAT programme (grant number ENV4 CT96-0314) and also EPSRC grant GR/L03088 *Combining Spatially Distributed Predictions from Neural Networks*.

## Footnotes

*To whom correspondence should be addressed.

[1]See  `http://www.ncrg.aston.ac.uk/Projects/NEUROSAT/NEUROSAT.html`

[4]The model allows small discontinuities in wind speed, which are consistent with frontal dynamics.

[5]This is equivalent to $n_1 = 24$ and $n_2 = 400$ for a scalar GP.

[6]Available from `http://www.ncrg.aston.ac.uk/netlab/index.html`.

## References

Cornford, D. 1998. Flexible Gaussian Process Wind Field Models. Technical Report NCRG/98/017, Neural Computing Research Group, Aston University, Aston Triangle, Birmingham, UK.

Cowles, M. K. and B. P. Carlin 1996. Markov-Chain Monte-Carlo Convergence Diagnostics—A Comparative Review. *Journal of the American Statistical Association* **91**, 883–904.

Cressie, N. A. C. 1993. *Statistics for Spatial Data*. New York: John Wiley and Sons.

Daley, R. 1991. *Atmospheric Data Analysis*. Cambridge: Cambridge University Press.

Handcock, M. S. and J. R. Wallis 1994. An Approach to Statistical Spatio-Temporal Modelling of Meteorological Fields. *Journal of the American Statistical Association* **89**, 368–378.

Neal, R. M. 1993. Probabilistic Inference Using Markov Chain Monte Carlo Methods. Technical Report CRG-TR-93-1, Department of Computer Science, University of Toronto. URL: `http://www.cs.utoronto.ca/~radford`.